# Predictive Sequence Learning in Recurrent Neocortical Circuits*

**R. P. N. Rao**
Computational Neurobiology Lab and
Sloan Center for Theoretical Neurobiology
The Salk Institute, La Jolla, CA 92037
rao@salk.edu

**T. J. Sejnowski**
Computational Neurobiology Lab and
Howard Hughes Medical Institute
The Salk Institute, La Jolla, CA 92037
terry@salk.edu

## Abstract

Neocortical circuits are dominated by massive excitatory feedback: more than eighty percent of the synapses made by excitatory cortical neurons are onto other excitatory cortical neurons. Why is there such massive recurrent excitation in the neocortex and what is its role in cortical computation? Recent neurophysiological experiments have shown that the plasticity of recurrent neocortical synapses is governed by a temporally asymmetric Hebbian learning rule. We describe how such a rule may allow the cortex to modify recurrent synapses for prediction of input sequences. The goal is to predict the next cortical input from the recent past based on previous experience of similar input sequences. We show that a temporal difference learning rule for prediction used in conjunction with dendritic back-propagating action potentials reproduces the temporally asymmetric Hebbian plasticity observed physiologically. Biophysical simulations demonstrate that a network of cortical neurons can learn to predict moving stimuli and develop direction selective responses as a consequence of learning. The space-time response properties of model neurons are shown to be similar to those of direction selective cells in alert monkey V1.

## 1 INTRODUCTION

The neocortex is characterized by an extensive system of recurrent excitatory connections between neurons in a given area. The precise computational function of this massive recurrent excitation remains unknown. Previous modeling studies have suggested a role for excitatory feedback in amplifying feedforward inputs [1]. Recently, however, it has been shown that recurrent excitatory connections between cortical neurons are modified according to a temporally asymmetric Hebbian learning rule: synapses that are activated slightly before the cell fires are strengthened whereas those that are activated slightly after are weakened [2, 3]. Information regarding the postsynaptic activity of the cell is conveyed back to the dendritic locations of synapses by back-propagating action potentials from the soma.

In this paper, we explore the hypothesis that recurrent excitation subserves the function of prediction and generation of temporal sequences in neocortical circuits [4, 5, 6]. We show

that a temporal difference based learning rule for prediction applied to backpropagating action potentials reproduces the experimentally observed phenomenon of asymmetric Hebbian plasticity. We then show that such a learning mechanism can be used to learn temporal sequences and the property of direction selectivity emerges as a consequence of learning to predict moving stimuli. Space-time response plots of model neurons are shown to be similar to those of direction selective cells in alert macaque V1.

## 2 TEMPORALLY ASYMMETRIC HEBBIAN PLASTICITY AND TEMPORAL DIFFERENCE LEARNING

To accurately predict input sequences, the recurrent excitatory connections in a network need to be adjusted such that the appropriate set of neurons are activated at each time step. This can be achieved by using a "temporal-difference" (TD) learning rule [5, 7]. In this paradigm of synaptic plasticity, an activated synapse is strengthened or weakened based on whether the difference between two temporally-separated predictions is positive or negative. This minimizes the errors in prediction by ensuring that the prediction generated by the neuron after synaptic modification is closer to the desired value than before (see [7] for more details).

In order to ascertain whether temporally-asymmetric Hebbian learning in cortical neurons can be interpreted as a form of temporal-difference learning, we used a two-compartment model of a cortical neuron consisting of a dendrite and a soma-axon compartment. The compartmental model was based on a previous study that demonstrated the ability of such a model to reproduce a range of cortical response properties [8]. The presence of voltage-activated sodium channels in the dendrite allowed back-propagation of action potentials from the soma into the dendrite. To study plasticity, excitatory postsynaptic potentials (EPSPs) were elicited at different time delays with respect to postsynaptic spiking by presynaptic activation of a single excitatory synapse located on the dendrite. Synaptic currents were calculated using a kinetic model of synaptic transmission with model parameters fitted to whole-cell recorded AMPA currents (see [9] for more details). Synaptic plasticity was simulated by incrementing or decrementing the value for maximal synaptic conductance by an amount proportional to the temporal-difference in the postsynaptic membrane potential at time instants $t + \Delta t$ and $t - \Delta t$ for presynaptic activation at time $t$. The delay parameter $\Delta t$ was set to 5 ms to yield results consistent with previous physiological experiments [2]. Presynaptic input to the model neuron was paired with postsynaptic spiking by injecting a depolarizing current pulse (10 ms, 200 pA) into the soma. Changes in synaptic efficacy were monitored by applying a test stimulus before and after pairing, and recording the EPSP evoked by the test stimulus.

Figure 1A shows the results of pairings in which the postsynaptic spike was triggered 5 ms after and 5 ms before the onset of the EPSP respectively. While the peak EPSP amplitude was increased 58.5% in the former case, it was decreased 49.4% in the latter case, qualitatively similar to experimental observations [2]. The critical window for synaptic modifications in the model depends on the parameter $\Delta t$ as well as the shape of the back-propagating action potential. This window of plasticity was examined by varying the time interval between presynaptic stimulation and postsynaptic spiking (with $\Delta t = 5$ ms). As shown in Figure 1B, changes in synaptic efficacy exhibited a highly asymmetric dependence on spike timing similar to physiological data [2]. Potentiation was observed for EPSPs that occurred between 1 and 12 ms before the postsynaptic spike, with maximal potentiation at 6 ms. Maximal depression was observed for EPSPs occurring 6 ms after the peak of the postsynaptic spike and this depression gradually decreased, approaching zero for delays greater than 10 ms. As in rat neocortical neurons, *Xenopus* tectal neurons, and cultured hippocampal neurons (see [2]), a narrow transition zone (roughly 3 ms in the model) separated the potentiation and depression windows.

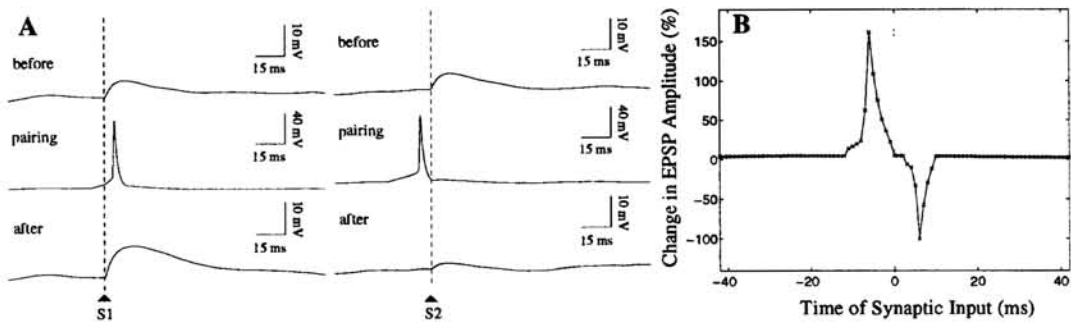

Figure 1: **Synaptic Plasticity in a Model Neocortical Neuron.** (**A**) (Left Panel) EPSP in the model neuron evoked by a presynaptic spike (S1) at an excitatory synapse ("before"). Pairing this presynaptic spike with postsynaptic spiking after a 5 ms delay ("pairing") induces long-term potentiation ("after"). (Right Panel) If presynaptic stimulation (S2) occurs 5 ms after postsynaptic firing, the synapse is weakened resulting in a corresponding decrease in peak EPSP amplitude. (**B**) Critical window for synaptic plasticity obtained by varying the delay between pre- and postsynaptic spiking (negative delays refer to presynaptic before postsynaptic spiking).

## 3   RESULTS

### 3.1   Learning Sequences using Temporally Asymmetric Hebbian Plasticity

To see how a network of model neurons can learn sequences using the learning mechanism described above, consider the simplest case of two excitatory neurons N1 and N2 connected to each other, receiving inputs from two separate input neurons I1 and I2 (Figure 2A). Suppose input neuron I1 fires before input neuron I2, causing neuron N1 to fire (Figure 2B). The spike from N1 results in a sub-threshold EPSP in N2 due to the synapse S2. If input arrives from I2 any time between 1 and 12 ms after this EPSP and the temporal summation of these two EPSPs causes N2 to fire, the synapse S2 will be strengthened. The synapse S1, on the other hand, will be weakened because the EPSP due to N2 arrives a few milliseconds after N1 has fired. Thus, on a subsequent trial, when input I1 causes neuron N1 to fire, N1 in turn causes N2 to fire several milliseconds *before* input I2 occurs due to the potentiation of the recurrent synapse S2 in previous trial(s) (Figure 2C). Input neuron I2 can thus be inhibited by the predictive feedback from N2 just before the occurrence of imminent input activity (marked by an asterisk in Figure 2C). This inhibition prevents input I2 from further exciting N2. Similarly, a positive feedback loop between neurons N1 and N2 is avoided because the synapse S1 was weakened in previous trial(s) (see arrows in Figures 2B and 2C). Figure 2D depicts the process of potentiation and depression of the two synapses as a function of the number of exposures to the I1-I2 input sequence. The decrease in latency of the predictive spike elicited in N2 with respect to the timing of input I2 is shown in Figure 2E. Notice that before learning, the spike occurs 3.2 ms after the occurrence of the input whereas after learning, it occurs 7.7 ms before the input.

### 3.2   Emergence of Direction Selectivity

In a second set of simulations, we used a network of recurrently connected excitatory neurons as shown in Figure 3A receiving retinotopic sensory input consisting of moving pulses of excitation (8 ms pulse of excitation at each neuron) in the rightward and leftward directions. The task of the network was to predict the sensory input by learning appropriate recurrent connections such that a given neuron in the network starts firing several milliseconds before the arrival of its input pulse of excitation. The network was comprised of two parallel chains of neurons with mutual inhibition (dark arrows) between corresponding pairs of neurons along the two chains. The network was initialized such that within a chain, a given

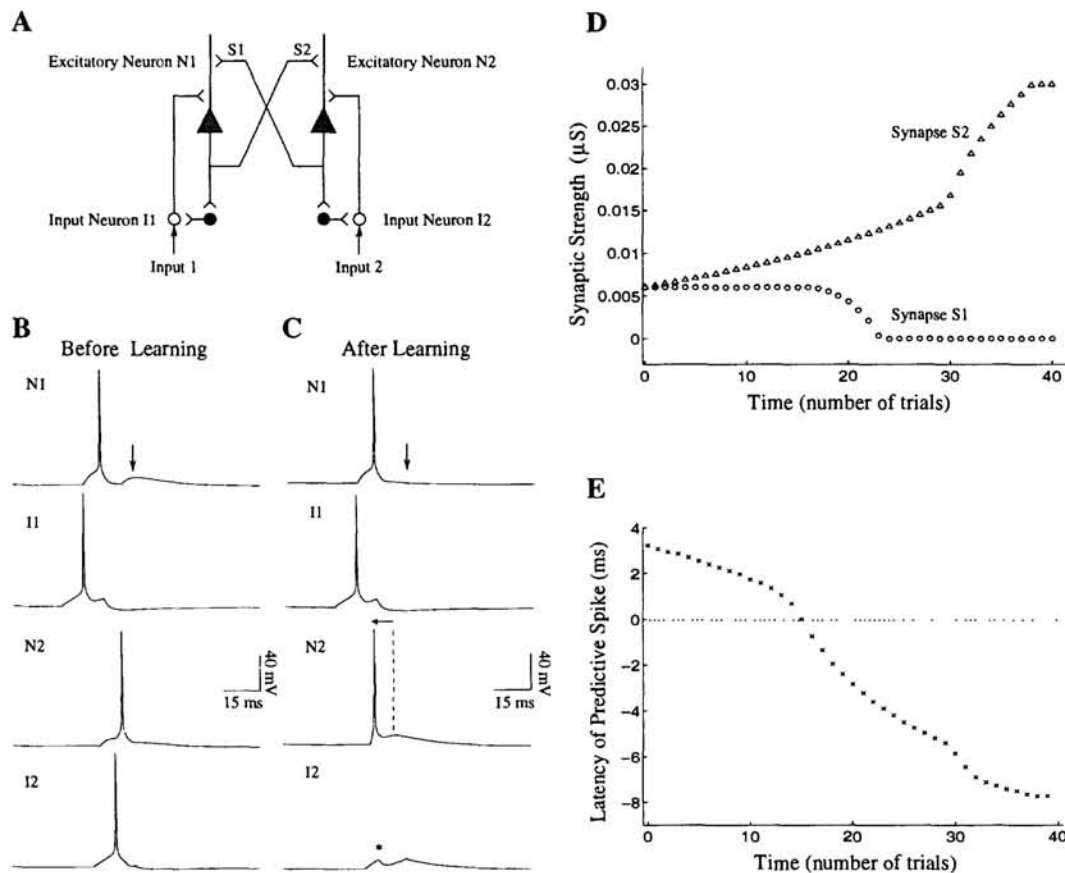

Figure 2: **Learning to Predict using Temporally Asymmetric Hebbian Learning**. (A) Network of two model neurons N1 and N2 recurrently connected via excitatory synapses S1 and S2, with input neurons I1 and I2. N1 and N2 inhibit the input neurons via inhibitory interneurons (darkened circles). (B) Network activity elicited by the sequence I1 followed by I2. (C) Network activity for the same sequence after 40 trials of learning. Due to strengthening of recurrent synapse S2, recurrent excitation from N1 now causes N2 to fire several ms before the expected arrival of input I2 (dashed line), allowing it to inhibit I2 (asterisk). Synapse S1 has been weakened, preventing re-excitation of N1 (downward arrows show decrease in EPSP). (D) Potentiation and depression of synapses S1 and S2 respectively during the course of learning. Synaptic strength was defined as maximal synaptic conductance in the kinetic model of synaptic transmission [9]. (E) Latency of predictive spike in N2 during the course of learning measured with respect to the time of input spike in I2 (dotted line).

excitatory neuron received both excitation and inhibition from its predecessors and successors (Figure 3B). Excitatory and inhibitory synaptic currents were calculated using kinetic models of synaptic transmission based on properties of AMPA and $GABA_A$ receptors as determined from whole-cell recordings [9]. Maximum conductances for all synapses were initialized to small positive values (dotted lines in Figure 3C) with a slight asymmetry in the recurrent excitatory connections for breaking symmetry between the two chains.

The network was exposed alternately to leftward and rightward moving stimuli for a total of 100 trials. The excitatory connections (labeled 'EXC' in Figure 3B) were modified according to the asymmetric Hebbian learning rule in Figure 1B while the excitatory connections onto the inhibitory interneuron (labeled 'INH') were modified according to an asymmetric anti-Hebbian learning rule that reversed the polarity of the rule in Figure 1B. The synaptic conductances learned by two neurons (marked N1 and N2 in Figure 3A) located at corresponding positions in the two chains after 100 trials of exposure to the moving stimuli are shown in Figure 3C (solid line). Initially, for rightward motion, the slight asymmetry in

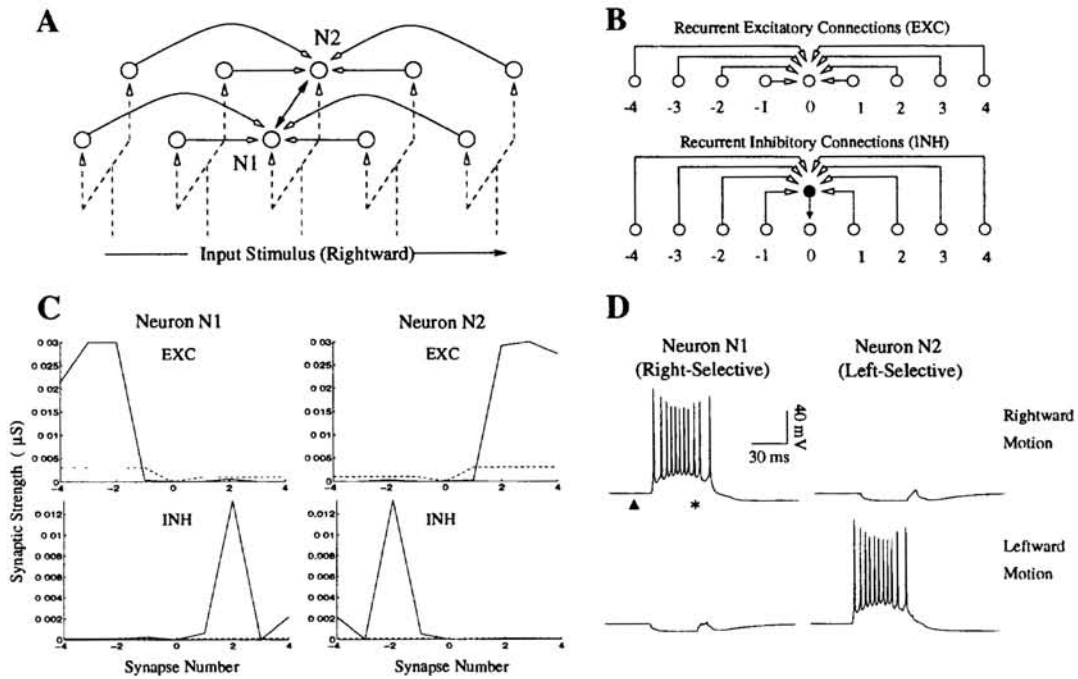

Figure 3: **Direction Selectivity in the Model**. (**A**) A model network consisting of two chains of recurrently connected neurons receiving retinotopic inputs. A given neuron receives recurrent excitation and recurrent inhibition (white-headed arrows) as well as inhibition (dark-headed arrows) from its counterpart in the other chain. (**B**) Recurrent connections to a given neuron (labeled '0') arise from 4 preceding and 4 succeeding neurons in its chain. Inhibition at a given neuron is mediated via a GABAergic interneuron (darkened circle). (**C**) Synaptic strength of recurrent excitatory (EXC) and inhibitory (INH) connections to neurons N1 and N2 before (dotted lines) and after learning (solid lines). Synapses were adapted during 100 trials of exposure to alternating leftward and rightward moving stimuli. (**D**) Responses of neurons N1 and N2 to rightward and leftward moving stimuli. As a result of learning, neuron N1 has become selective for rightward motion (as have other neurons in the same chain) while neuron N2 has become selective for leftward motion. In the preferred direction, each neuron starts firing several milliseconds before the actual input arrives at its soma (marked by an asterisk) due to recurrent excitation from preceding neurons. The dark triangle represents the start of input stimulation in the network.

the initial excitatory connections of neuron N1 allows it to fire slightly earlier than neuron N2 thereby inhibiting neuron N2. Additionally, since the EPSPs from neurons lying on the left of N1 occur before N1 fires, the excitatory synapses from these neurons are strengthened while the excitatory synapses from these same neurons to the inhibitory interneuron are weakened according to the two learning rules mentioned above. On the other hand, the excitatory synapses from neurons lying on the right side of N1 are weakened while inhibitory connections are strengthened since the EPSPs due to these connections occur after N1 has fired. The synapses on neuron N2 and its associated interneuron remain unaltered since there is no postsynaptic firing (due to inhibition by N1) and hence no back-propagating action potentials in the dendrite. As shown in Figure 3C, after 100 trials, the excitatory and inhibitory connections to neuron N1 exhibit a marked asymmetry, with excitation originating from neurons on the left and inhibition from neurons on the right. Neuron N2 exhibits the opposite pattern of connectivity. As expected, neuron N1 was found to be selective for rightward motion while neuron N2 was selective for leftward motion (Figure 3D). Moreover, when stimulus motion is in the preferred direction, each neuron starts firing several milliseconds before the time of arrival of the input stimulus at its soma (marked by an asterisk) due to recurrent excitation from preceding neurons. Conversely, motion in the nonpreferred direction triggers recurrent inhibition from preceding neurons as well as inhibition

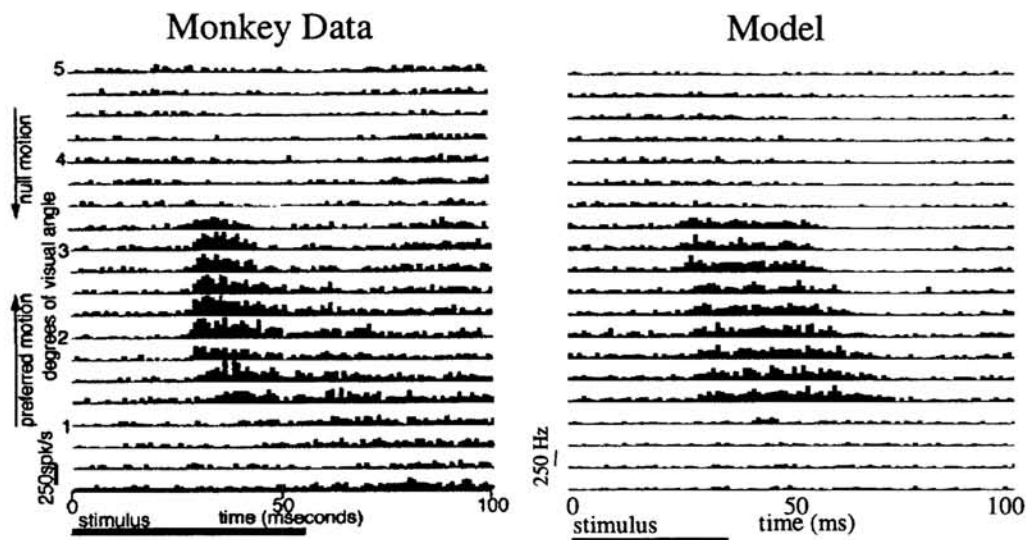

Figure 4: **Comparison of Monkey and Model Space-Time Response Plots**. (Left) Sequence of PSTHs obtained by flashing optimally oriented bars at 20 positions across the 5°-wide receptive field (RF) of a complex cell in alert monkey V1 (from [11]). The cell's preferred direction is from the part of the RF represented at the bottom towards the top. Flash duration = 56 ms; inter-stimulus delay = 100 ms; 75 stimulus presentations. (Right) PSTHs obtained from a model neuron after stimulating the chain of neurons at 20 positions to the left and right side of the given neuron. Lower PSTHs represent stimulations on the preferred side while upper PSTHs represent stimulations on the null side.

from the active neuron in the corresponding position in the other chain. Thus, the learned pattern of connectivity allows the direction selective neurons comprising the two chains in the network to conjointly code for and predict the moving input stimulus in each direction. The average firing rate of neurons in the network for the preferred direction was 75.7 Hz, which is in the range of cortical firing rates for moving bar stimuli. Assuming a 200 $\mu$m separation between excitatory model neurons in each chain and utilizing known values for the cortical magnification factor in monkey striate cortex, one can estimate the preferred stimulus velocity of model neurons to be 3.1°/s in the fovea and 27.9°/s in the periphery (at an eccentricity of 8°). Both of these values fall within the range of monkey striate cortical velocity preferences [11].

The model predicts that the neuroanatomical connections for a direction selective neuron should exhibit a pattern of asymmetrical excitation and inhibition similar to Figure 3C. A recent study of direction selective cells in awake monkey V1 found excitation on the preferred side of the receptive field and inhibition on the null side consistent with the pattern of connections learned by the model [11]. For comparison with this experimental data, spontaneous background activity in the model was generated by incorporating Poisson-distributed random excitatory and inhibitory alpha synapses on the dendrite of each model neuron. Post stimulus time histograms (PSTHs) and space-time response plots were obtained by flashing optimally oriented bar stimuli at random positions in the cell's activating region. As shown in Figure 4, there is good qualitative agreement between the response plot for a complex cell and that for the model. Both space-time plots show a progressive shortening of response onset time and an increase in response transiency going in the preferred direction: in the model, this is due to recurrent excitation from progressively closer cells on the preferred side. Firing is reduced to below background rates 40-60 ms after stimulus onset in the upper part of the plots: in the model, this is due to recurrent inhibition from cells on the null side. The response transiency and shortening of response time course appears as a slant in the space-time maps, which can be related to the neuron's velocity sensitivity [11].

## 4  CONCLUSIONS

Our results show that a network of recurrently connected neurons endowed with a temporal-difference based asymmetric Hebbian learning mechanism can learn a predictive model of its spatiotemporal inputs. When exposed to moving stimuli, neurons in a simulated network learned to fire several milliseconds before the expected arrival of an input stimulus and developed direction selectivity as a consequence of learning. The model predicts that a direction selective neuron should start responding several milliseconds before the preferred stimulus enters its retinal input dendritic field (such predictive neural activity has recently been reported in retinal ganglion cells [10]). Temporally asymmetric Hebbian learning has previously been suggested as a possible mechanism for sequence learning in the hippocampus [4] and as an explanation for the asymmetric expansion of hippocampal place fields during route learning [12]. Some of these theories require relatively long temporal windows of synaptic plasticity (on the order of several hundreds of milliseconds) [4] while others have utilized temporal windows in the millisecond range for coincidence detection [3]. Sequence learning in our model is based on a window of plasticity in the 10 to 15 ms range which is roughly consistent with recent physiological observations [2] (see also [13]). The idea that prediction and sequence learning may constitute an important goal of the neocortex has previously been suggested in the context of statistical and information theoretic models of cortical processing [4, 5, 6]. Our biophysical simulations suggest a possible implementation of such models in cortical circuitry. Given the universality of the problem of encoding and generating temporal sequences in both sensory and motor domains, the hypothesis of predictive sequence learning in recurrent neocortical circuits may help provide a unifying principle for studying cortical structure and function.

## Footnotes

*This research was supported by the Sloan Foundation and Howard Hughes Medical Institute.

## References

[1] R. J. Douglas *et al.*, *Science* **269**, 981 (1995); H. Suarez *et al.*, *J. Neurosci.* **15**, 6700 (1995); R. Maex and G. A. Orban, *J. Neurophysiol.* **75**, 1515 (1996); P. Mineiro and D. Zipser, *Neural Comput.* **10**, 353 (1998); F. S. Chance *et al.*, *Nature Neuroscience* **2**, 277 (1999).

[2] H. Markram *et al.*, *Science* **275**, 213 (1997); W. B. Levy and O. Steward, *Neuroscience* **8**, 791 (1983); D. Debanne *et al.*, *Proc. Natl. Acad. Sci. U.S.A.* **91**, 1148 (1994); L. I. Zhang *et al.*, *Nature* **395**, 37 (1998); G. Q. Bi and M. M. Poo, *J. Neurosci.* **18**, 10464 (1998).

[3] W. Gerstner *et al.*, *Nature* **383**, 76 (1996); R. Kempter *et al.*, in *Advances in Neural Info. Proc. Systems 11*, M. S. Kearns, S. A. Solla and D. A. Cohn, Eds. (MIT Press, Cambridge, MA, 1999), pp. 125–131.

[4] L. F. Abbott and K. I. Blum, *Cereb. Cortex* **6**, 406 (1996); W. Gerstner and L. F. Abbott, *J. Comput. Neurosci.* **4**, 79 (1997); A. A. Minai and W. B. Levy, in *Proceedings of the 1993 World Congress on Neural Networks* II, 505 (1993).

[5] P. R. Montague and T. J. Sejnowski, *Learning and Memory* **1**, 1 (1994); P. R. Montague *et al.*, *Nature* **377**, 725 (1995); W. Schultz *et al.*, *Science* **275**, 1593 (1997).

[6] R. P. N. Rao and D. H. Ballard, *Neural Computation* **9**, 721 (1997); R. P. N. Rao and D. H. Ballard, *Nature Neuroscience* **2**, 79 (1999); H. Barlow, *Perception* **27**, 885 (1998).

[7] R. S. Sutton, *Machine Learning* **3**, 9 (1988); R. S. Sutton and A. G. Barto, in *Learning and Computational Neuroscience: Foundations of Adaptive Networks*, M. Gabriel and J. W. Moore, editors (MIT Press, Cambridge, MA, 1990).

[8] Z. F. Mainen and T. J. Sejnowski, *Nature* **382**, 363 (1996).

[9] A. Destexhe *et al.*, in *Methods in Neuronal Modeling*, C. Koch and I. Segev, editors, (MIT Press, Cambridge, MA, 1998).

[10] M. J. Berry *et al.*, *Nature* **398**, 334 (1999).

[11] M. S. Livingstone, *Neuron* **20**, 509 (1998).

[12] M. R. Mehta *et al.*, *Proc. Natl. Acad. Sci. U.S.A.* **94**, 8918 (1997).

[13] L. F. Abbott and S. Song, in *Advances in Neural Info. Proc. Systems 11*, M. S. Kearns, S. A. Solla and D. A. Cohn, Eds. (MIT Press, Cambridge, MA, 1999), pp. 69–75.